# Spectral Bounds for Sparse PCA:
# Exact and Greedy Algorithms

**Baback Moghaddam**
MERL
Cambridge MA, USA
*baback@merl.com*

**Yair Weiss**
Hebrew University
Jerusalem, Israel
*yweiss@cs.huji.ac.il*

**Shai Avidan**
MERL
Cambridge MA, USA
*avidan@merl.com*

## Abstract

Sparse PCA seeks *approximate* sparse "eigenvectors" whose projections capture the maximal variance of data. As a cardinality-constrained and *non-convex* optimization problem, it is NP-hard and is encountered in a wide range of applied fields, from bio-informatics to finance. Recent progress has focused mainly on continuous approximation and convex relaxation of the hard cardinality constraint. In contrast, we consider an alternative *discrete* spectral formulation based on variational eigenvalue bounds and provide an effective greedy strategy as well as provably *optimal* solutions using branch-and-bound search. Moreover, the exact methodology used reveals a simple renormalization step that improves approximate solutions obtained by *any* continuous method. The resulting performance gain of discrete algorithms is demonstrated on real-world benchmark data and in extensive Monte Carlo evaluation trials.

## 1 Introduction

PCA is indispensable as a basic tool for factor analysis and modeling of data. But despite its power and popularity, one key drawback is its lack of sparseness (*i.e.,* factor loadings are linear combinations of *all* the input variables). Yet sparse representations are generally desirable since they aid human understanding (*e.g.,* with gene expression data), reduce computational costs and promote better generalization in learning algorithms. In machine learning, input sparseness is closely related to feature selection and automatic relevance determination, problems of enduring interest to the learning community.

The earliest attempts at "sparsifying" PCA in the statistics literature consisted of simple axis rotations and component thresholding [1] with the underlying goal being essentially that of subset selection, often based on the identification of *principal variables* [8]. The first true computational technique, called SCoTLASS by Jolliffe & Uddin [6], provided a proper optimization framework using Lasso [12] but it proved to be computationally impractical. Recently, Zou *et al.* [14] proposed an elegant algorithm (SPCA) using their "Elastic Net" framework for $L_1$-penalized regression on regular PCs, solved very efficiently using *least angle regression* (LARS). Subsequently, d'Aspremont *et al.* [3] relaxed the "hard" cardinality constraint and solved for a *convex* approximation using semi-definite programming (SDP). Their "direct" formulation for sparse PCA (called DSCPA) has yielded promising results that are comparable to (if not better than) Zou *et al.*'s Lasso-based method, as demonstrated on the standard "Pit Props" benchmark dataset, known in the statistics community for its lack of sparseness and subsequent difficulty of interpretation.

We pursued an alternative approach using a spectral formulation based on the variational principle of the Courant-Fischer "Min-Max" theorem for solving maximal eigenvalue problems in dimensionality-constrained subspaces. By its very nature, the discrete view leads to a simple post-processing (renormalization) step that improves *any* approximate solution (*e.g.,* those given in [6, 14, 3]), and also provides bounds on (sub)optimality. More importantly, it points the way towards *exact* and provably *optimal* solutions using branch-and-bound search [9]. Our exact computational strategy parallels that of Ko *et al.* [7] who solved a different optimization problem (maximizing entropy with bounds on *determinants*). In the experiments we demonstrate the power of greedy and exact algorithms by first solving for the optimal sparse factors of the real-world "Pit Props" data, a *de facto* benchmark used by [6, 14, 3], and then present summary findings from a large comparative study using extensive Monte Carlo evaluation of the leading algorithms.

## 2  Sparse PCA Formulation

Sparse PCA can be cast as a cardinality-constrained quadratic program (QP): given a symmetric positive-definite (covariance) matrix $A \in \mathcal{S}_+^n$, maximize the quadratic form $x'Ax$ (variance) with a *sparse* vector $x \in \mathcal{R}^n$ having no more than $k$ non-zero elements:

$$\begin{aligned} \mathbf{max} \quad & x'A\,x & (1)\\ \text{subject to} \quad & x'x = 1 \\ & \mathrm{card}(x) \le k \end{aligned}$$

where $\mathrm{card}(x)$ denotes the $L_0$ norm. This optimization problem is non-convex, NP-hard and therefore *intractable*. Assuming we can solve for the optimal vector $\hat{x}$, subsequent sparse factors can be obtained using recursive *deflation* of $A$, as in standard numerical routines. The sparseness is controlled by the value(s) of $k$ (in different factors) and can be viewed as a design parameter or as an unknown quantity itself (known only to the oracle). Alas, there are currently no guidelines for setting $k$, especially with multiple factors (*e.g.,* orthogonality is often relaxed) and unlike ordinary PCA some decompositions may not be unique.[1] Indeed, one of the contributions of this paper is in providing a sound theoretical basis for selecting $k$, thus clarifying the "art" of crafting sparse PCA factors.

Note that without the cardinality constraint, the quadratic form in Eq.(1) is a Rayleigh-Ritz quotient obeying the analytic bounds $\lambda_{\min}(A) \le x'Ax/x'x \le \lambda_{\max}(A)$ with corresponding unique eigenvector solutions. Therefore, the optimal objective value (variance) is simply the maximum eigenvalue $\lambda_n(A)$ of the principal eigenvector $\hat{x} = u_n$ — **Note:** throughout the paper the rank of all $(\lambda_i, u_i)$ is in *increasing* order of magnitude, hence $\lambda_{\min} = \lambda_1$ and $\lambda_{\max} = \lambda_n$. With the (nonlinear) cardinality constraint however, the optimal objective value is strictly less than $\lambda_{\max}(A)$ for $k < n$ and the principal eigenvectors are no longer instrumental in the solution. Nevertheless, we will show that the *eigenvalues* of $A$ continue to play a key role in the analysis and design of exact algorithms.

### 2.1  Optimality Conditions

First, let us consider what conditions *must* be true if the oracle revealed the optimal solution to us: a unit-norm vector $\hat{x}$ with cardinality $k$ yielding the *maximum* objective value $v^*$. This would necessarily imply that $\hat{x}'A\,\hat{x} = z'A_k z$ where $z \in \mathcal{R}^k$ contains the same $k$ non-zero elements in $\hat{x}$ and $A_k$ is the $k \times k$ principal submatrix of $A$ obtained by deleting the rows and columns corresponding to the zero indices of $\hat{x}$ (or equivalently, by extracting the rows and columns of non-zero indices). Like $\hat{x}$, the $k$-vector $z$ will be unit norm and $z'A_k z$ is then equivalent to a standard *unconstrained* Rayleigh-Ritz quotient. Since this subproblem's maximum variance is $\lambda_{\max}(A_k)$, then this *must* be the optimal objective $v^*$. We will now summarize this important observation with the following proposition.

**Proposition 1**. *The optimal value $v^*$ of the sparse PCA optimization problem in Eq.(1) is equal to $\lambda_{\max}(A_k^*)$, where $A_k^*$ is the $k \times k$ principal submatrix of $A$ with the largest maximal eigenvalue. In particular, the non-zero elements of the optimal sparse factor $\hat{x}$ are exactly equal to the elements of $u_k^*$, the principal eigenvector of $A_k^*$.*

This underscores the inherent combinatorial nature of sparse PCA and the equivalent class of cardinality-constrained optimization problems. However, despite providing an exact formulation and revealing necessary conditions for optimality (and in such simple matrix terms), this proposition does not suggest an efficient method for actually *finding* the principal submatrix $A_k^*$ — short of an enumerative exhaustive search, which is impractical for $n > 30$ due to the exponential growth of possible submatrices. Still, exhaustive search is a viable method for small $n$ which guarantees optimality for "toy problems" and small real-world datasets, thus calibrating the *quality* of approximations (via the optimality gap).

## 2.2 Variational Renormalization

Proposition 1 immediately suggests a rather simple but (as it turns out) quite effective computational "fix" for *improving* candidate sparse PC factors obtained by *any* continuous algorithm (*e.g.,* the various solutions found in [6, 14, 3]).

**Proposition 2**. *Let $\tilde{x}$ be a unit-norm candidate factor with cardinality $k$ as found by any (approximation) technique. Let $\tilde{z}$ be the non-zero subvector of $\tilde{x}$ and $u_k$ be the principal (maximum) eigenvector of the submatrix $A_k$ defined by the same non-zero indices of $\tilde{x}$. If $\tilde{z} \neq u_k(A_k)$, then $\tilde{x}$ is* not *the optimal solution. Nevertheless, by replacing $\tilde{x}$'s nonzero elements with those of $u_k$ we guarantee an* increase *in the variance, from $\tilde{v}$ to $\lambda_k(A_k)$.*

This variational renormalization suggests (somewhat ironically) that given a continuous (approximate) solution, it is almost certainly better to *discard* the loadings and keep only the sparsity pattern with which to solve the smaller *unconstrained* subproblem for the indicated submatrix $A_k$. This simple procedure (or "fix" as referred to herein) can *never* decrease the variance and will surely improve any continuous algorithm's performance.

In particular, the rather expedient but *ad-hoc* technique of "simple thresholding" (ST) [1] — *i.e.,* setting the $n - k$ *smallest* absolute value loadings of $u_n(A)$ to zero and then normalizing to unit-norm — is therefore *not* recommended for sparse PCA. In Section 3, we illustrate how this "straw-man" algorithm can be enhanced with proper renormalization. Consequently, past performance benchmarks using this simple technique may need revision — *e.g.,* previous results on the "Pit Props" dataset (Section 3). Indeed, most of the sparse PCA factors published in the literature can be readily improved (almost by inspection) with the proper renormalization, and at the mere cost of a single $k$-by-$k$ eigen-decomposition.

## 2.3 Eigenvalue Bounds

Recall that the objective value $v^*$ in Eq.(1) is bounded by the spectral radius $\lambda_{\max}(A)$ (by the Rayleigh-Ritz theorem). Furthermore, the spectrum of $A$'s principal submatrices was shown to play a key role in *defining* the optimal solution. Not surprisingly, the two eigenvalue spectra are related by an inequality known as the *Inclusion Principle*.

**Theorem 1** Inclusion Principle. *Let $A$ be a symmetric $n \times n$ matrix with spectrum $\lambda_i(A)$ and let $A_k$ be any $k \times k$ principal submatrix of $A$ for $1 \leq k \leq n$ with eigenvalues $\lambda_i(A_k)$. For each integer $i$ such that $1 \leq i \leq k$*

$$\lambda_i(A) \leq \lambda_i(A_k) \leq \lambda_{i+n-k}(A) \tag{2}$$

*Proof.* The proof, which we omit, is a rather straightforward consequence of imposing a sparsity pattern of cardinality $k$ as an additional orthogonality constraint in the variational inequality of the Courant-Fischer "Min-Max" theorem (see [13] for example).

In other words, the eigenvalues of a symmetric matrix form upper and lower bounds for the eigenvalues of all its principal submatrices. A special case of Eq.(2) with $k = n - 1$ leads to the well-known eigenvalue *interlacing property* of symmetric matrices:

$$\lambda_1(A_n) \leq \lambda_1(A_{n-1}) \leq \lambda_2(A_n) \leq \ldots \leq \lambda_{n-1}(A_n) \leq \lambda_{n-1}(A_{n-1}) \leq \lambda_n(A_n) \quad (3)$$

Hence, the spectra of $A_n$ and $A_{n-1}$ interleave or *interlace* each other, with the eigenvalues of the larger matrix "bracketing" those of the smaller one. Note that for *positive-definite* symmetric matrices (covariances), augmenting $A_m$ to $A_{m+1}$ (adding a new variable) will always *expand* the spectral range: reducing $\lambda_{\min}$ and increasing $\lambda_{\max}$. Thus for eigenvalue maximization, the inequality constraint $\mathrm{card}(x) \leq k$ in Eq.(1) is a tight *equality* at the optimum. Therefore, the maximum variance is achieved at the preset upper limit $k$ of cardinality. Moreover, the function $v^*(k)$, the optimal variance for a given cardinality, is monotone *increasing* with range $[\sigma^2_{\max}(A), \lambda_{\max}(A)]$, where $\sigma^2_{\max}$ is the largest diagonal element (variance) in $A$. Hence, a concise and informative way to quantify the performance of an algorithm is to plot its variance curve $\tilde{v}(k)$ and compare it with the optimal $v^*(k)$.

Since we seek to *maximize* variance, the relevant inclusion bound is obtained by setting $i = k$ in Eq.(2), which yields lower and upper bounds for $\lambda_k(A_k) = \lambda_{\max}(A_k)$,

$$\lambda_k(A) \leq \lambda_{\max}(A_k) \leq \lambda_{\max}(A) \quad (4)$$

This shows that the $k$-th *smallest* eigenvalue of $A$ is a *lower bound* for the maximum variance possible with cardinality $k$. The utility of this lower bound is in doing away with the "guesswork" (and the oracle) in setting $k$. Interestingly, we now see that the spectrum of $A$ which has traditionally guided the selection of eigenvectors for dimensionality reduction (*e.g.,* in classical PCA), can also be consulted in sparse PCA to help pick the cardinality required to capture the desired (minimum) variance. The lower bound $\lambda_k(A)$ is also useful for speeding up branch-and-bound search (see next Section). Note that if $\lambda_k(A)$ is close to $\lambda_{max}(A)$ then practically any principal submatrix $A_k$ can yield a near-optimal solution.

The right-hand inequality in Eq.(4) is a fixed (loose) upper bound $\lambda_{\max}(A)$ for *all* $k$. But in branch-and-bound search, any *intermediate* subproblem $A_m$, with $k \leq m \leq n$, yields a new and *tighter* bound $\lambda_{\max}(A_m)$ for the objective $v^*(k)$. Therefore, all bound computations are efficient and relatively inexpensive (*e.g.,* using the *power method*).

The inclusion principle also leads to some interesting constraints on *nested* submatrices. For example, among all $m$ possible $(m - 1)$-by-$(m - 1)$ principal submatrices of $A_m$, obtained by deleting the $j$-th row and column, there is at least one submatrix $A_{m-1} = A_{\setminus j}$ whose maximal eigenvalue is a major fraction of its parent (*e.g.,* see p. 189 in [4])

$$\exists\, j : \quad \lambda_{m-1}(A_{\setminus j}) \geq \frac{m-1}{m} \lambda_m(A_m) \quad (5)$$

The implication of this inequality for search algorithms is that it is simply not possible for the spectral radius of *every* submatrix $A_{\setminus j}$ to be arbitrarily small, especially for large $m$. Hence, with large matrices (or large cardinality) nearly all the variance $\lambda_n(A)$ is captured.

## 2.4 Combinatorial Optimization

Given Propositions 1 and 2, the inclusion principle, the interlacing property and especially the monotonic nature of the variance curves $v(k)$, a general class of (binary) *integer programming* (IP) optimization techniques [9] seem ideally suited for sparse PCA. Indeed, a greedy technique like *backward elimination* is already suggested by the bound in Eq.(5): start with the full index set $I = \{1, 2, \ldots, n\}$ and sequentially delete the variable $j$ which yields the maximum $\lambda_{\max}(A_{\setminus j})$ until only $k$ elements remain. However, for *small* cardinalities $k << n$, the computational cost of backward search can grow to near

maximum complexity $\approx O(n^4)$. Hence its counterpart *forward selection* is preferred: start with the null index set $I = \{\}$ and sequentially add the variable $j$ which yields the maximum $\lambda_{\max}(A_{+j})$ until $k$ elements are selected. Forward greedy search has *worst-case* complexity $< O(n^3)$. The best overall strategy for this problem was empirically found to be a *bi-directional* greedy search: run a forward pass (from 1 to $n$) plus a second (independent) backward pass (from $n$ to 1) and pick the better solution at each $k$. This proved to be remarkably effective under extensive Monte Carlo evaluation and with real-world datasets. We refer to this discrete algorithm as *greedy* sparse PCA or **GSPCA**.

Despite the expediency of near-optimal greedy search, it is nevertheless worthwhile to invest in optimal solution strategies, especially if the sparse PCA problem is in the application domain of finance or engineering, where even a small optimality gap can accrue substantial losses over time. As with Ko *et al.* [7], our branch-and-bound relies on computationally efficient bounds — in our case, the upper bound in Eq.(4), used on all active *sub*problems in a (FIFO) queue for *depth-first* search. The *lower* bound in Eq.(4) can be used to sort the queue for a more efficient *best-first* search [9]. This *exact* algorithm (referred to as **ESPCA**) is *guaranteed* to terminate with the optimal solution. Naturally, the search time depends on the quality (variance) of initial candidates. The solutions found by dual-pass greedy search (**GSPCA**) were found to be ideal for initializing **ESPCA**, as their quality was typically quite high. Note however, that even with good initializations, branch-and-bound search can take a *long* time (*e.g.* 1.5 hours for $n = 40$, $k = 20$). In practice, early termination with set thresholds based on eigenvalue bounds can be used.

In general, a cost-effective strategy that we can recommend is to first run GSPCA (or at least the forward pass) and then either settle for its (near-optimal) variance or else use it to initialize ESPCA for finding the optimal solution. A full GSPCA run has the added benefit of giving near-optimal solutions for *all* cardinalities at once, with run-times that are typically $O(10^2)$ faster than a *single* approximation with a continuous method.

## 3   Experiments

We evaluated the performance of GSPCA (and validated ESPCA) on various synthetic covariance matrices with $10 \leq n \leq 40$ as well as real-world datasets from the UCI ML repository with excellent results. We present few typical examples in order to illustrate the advantages and power of discrete algorithms. In particular, we compared our performance against 3 continuous techniques: simple thresholding (ST) [1], SPCA using an "Elastic Net" $L_1$-regression [14] and DSPCA using semidefinite programming [3].

We first revisited the "Pit Props" dataset [5] which has become a standard benchmark and a classic example of the difficulty of interpreting fully loaded factors with standard PCA. The first 6 ordinary PCs capture 87% of the total variance, so following the methodology in [3], we compared the explanatory power of our exact method (ESPCA) using 6 *sparse* PCs. Table 1 shows the first 3 PCs and their loadings. SPCA captures 75.8% of the variance with a cardinality pattern of 744111 (the $k$'s for the 6 PCs) thus totaling 18 non-zero loadings [14] whereas DSPCA captures 77.3% with a sparser cardinality pattern 623111 totaling 14 non-zero loadings [3]. We aimed for an even sparser 522111 pattern (with only 12 non-zero loadings) yet captured nearly the same variance: 75.9% — *i.e.,* more than SPCA with 18 loadings and slightly less than DSPCA with 14 loadings.

Using the evaluation protocol in [3], we compared the cumulative variance and cumulative cardinality with the published results of SPCA and DSPCA in Figure 1. Our goal was to match the explained variance but do so with a sparser representation. The ESPCA loadings in Table 1 are optimal under the definition given in Section 2. The run-time of ESPCA, *including* initialization with a bi-directional pass of GSPCA, was negligible for this dataset ($n = 13$). Computing each factor took less than 50 msec in Matlab 7.0 on a 3GHz P4.

| | $x_1$ | $x_2$ | $x_3$ | $x_4$ | $x_5$ | $x_6$ | $x_7$ | $x_8$ | $x_9$ | $x_{10}$ | $x_{11}$ | $x_{12}$ | $x_{13}$ |
|---|---|---|---|---|---|---|---|---|---|---|---|---|---|
| **SPCA** : PC1 | -.477 | -.476 | 0 | 0 | .177 | 0 | -.250 | -.344 | -.416 | -.400 | - | 0 | 0 |
| PC2 | 0 | 0 | .785 | .620 | 0 | 0 | 0 | -.021 | 0 | 0 | 0 | .013 | 0 |
| PC3 | 0 | 0 | 0 | 0 | .640 | .589 | .492 | 0 | 0 | 0 | 0 | 0 | -.015 |
| **DSPCA** : PC1 | -.560 | -.583 | 0 | 0 | 0 | 0 | -.263 | -.099 | -.371 | -.362 | 0 | 0 | 0 |
| PC2 | 0 | 0 | .707 | .707 | 0 | 0 | 0 | 0 | 0 | 0 | 0 | 0 | 0 |
| PC3 | 0 | 0 | 0 | 0 | 0 | -.793 | -.610 | 0 | 0 | 0 | 0 | 0 | .012 |
| **ESPCA** : PC1 | -.480 | -.491 | 0 | 0 | 0 | 0 | -.405 | 0 | -.423 | -.431 | 0 | 0 | 0 |
| PC2 | 0 | 0 | .707 | .707 | 0 | 0 | 0 | 0 | 0 | 0 | 0 | 0 | 0 |
| PC3 | 0 | 0 | 0 | 0 | 0 | -.814 | -.581 | 0 | 0 | 0 | 0 | 0 | 0 |

Table 1: Loadings for first 3 sparse PCs of the Pit Props data. See Figure 1(a) for plots of the corresponding cumulative variances. Original SPCA and DSPCA loadings taken from [14, 3].

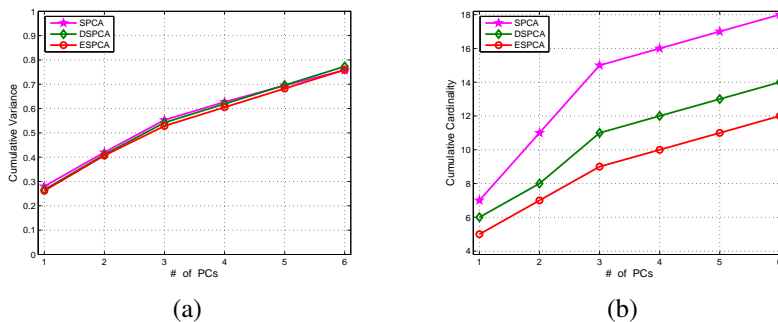

(a)                          (b)

Figure 1: Pit Props: **(a)** cumulative variance and **(b)** cumulative cardinality for first 6 sparse PCs. Sparsity patterns (cardinality $k_i$ for PC$_i$, with $i = 1, 2, \ldots, 6$) are 744111 for SPCA (magenta $\star$), 623111 for DSPCA (green $\diamond$) and an optimal 522111 for ESPCA (red $\circ$). The factor loadings for the first 3 sparse PCs are shown in Table 1. Original SPCA and DSPCA results taken from [14, 3].

To specifically demonstrate the benefits of the variational renormalization of Section 2.2, consider SPCA's first sparse factor in Table 1 (the 1st row of SPCA block) found by iterative ($L_1$-penalized) optimization and unit-norm scaling. It captures 28% of the total data variance, but after the variational renormalization the variance *increases* to 29%. Similarly, the first sparse factor of DSPCA in Table 1 (1st row of DSPCA block) captures 26.6% of the total variance, whereas after variational renormalization it captures 29% — a gain of 2.4% for the mere additional cost of a 7-by-7 eigen-decomposition. Given that variational renormalization results in the maximum variance possible for the indicated sparsity pattern, omitting such a simple post-processing step is counter-productive, since otherwise the approximations would be, in a sense, *doubly* sub-optimal: both globally and "locally" in the subspace (subset) of the sparsity pattern found.

We now give a representative summary of our extensive Monte Carlo (MC) evaluation of GSPCA and the 3 continuous algorithms. To show the most typical or average-case performance, we present results with random covariance matrices from synthetic stochastic Brownian processes of various degrees of smoothness, ranging from sub-Gaussian to super-Gaussian. Every MC run consisted of 50,000 covariance matrices and the (normalized) variance curves $\tilde{v}(k)$. For each matrix, **E**SPCA was used to find the *optimal* solution as "ground truth" for subsequent calibration, analysis and performance evaluation.

For SPCA we used the LARS-based "Elastic Net" SPCA Matlab toolbox of Sjöstrand [10] which is equivalent to Zou *et al.*'s SPCA source code, which is also freely available in R. For DSPCA we used the authors' own Matlab source code [2] which uses the SDP toolbox SeDuMi1.0x [11]. The main DSPCA routine *PrimalDec*$(A, k)$ was called with $k-1$ instead of $k$, for all $k > 2$, as per the recommended calibration (see documentation in [3, 2]).

In our MC evaluations, all continuous methods (ST, SPCA and DSPCA) had variational renormalization post-processing (applied to their the "declared" solution). Note that comparing GSPCA with the *raw* output of these algorithms would be rather pointless, since

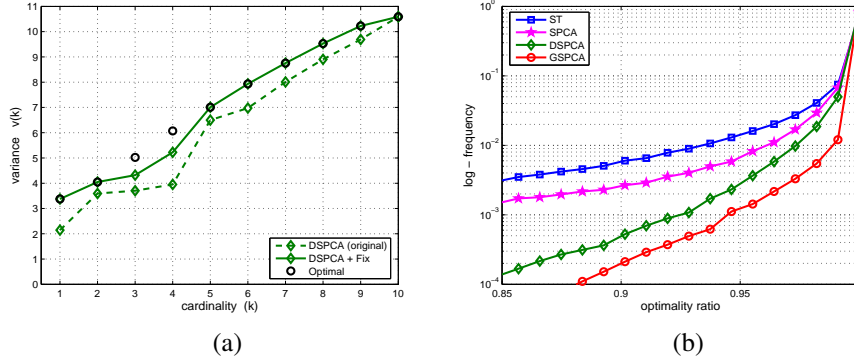

|     |     |
|:---:|:---:|
| (a) | (b) |

Figure 2: **(a)** Typical variance curve $v(k)$ for a continuous algorithm *without* post-processing (original: dash green) and *with* variational renormalization (+ Fix: solid green). Optimal variance (black ∘) by **ESPCA**. At $k = 4$ optimality ratio *increases* from 0.65 to 0.86 (a 21% gain). **(b)** Monte Carlo study: log-likelihood of optimality ratio at max-complexity ($k = 8$, $n = 16$) for ST (blue []), DSPCA (green ⋄), SPCA (magenta ⋆) and GSPCA (red ∘). Continuous methods were "fixed" in (b).

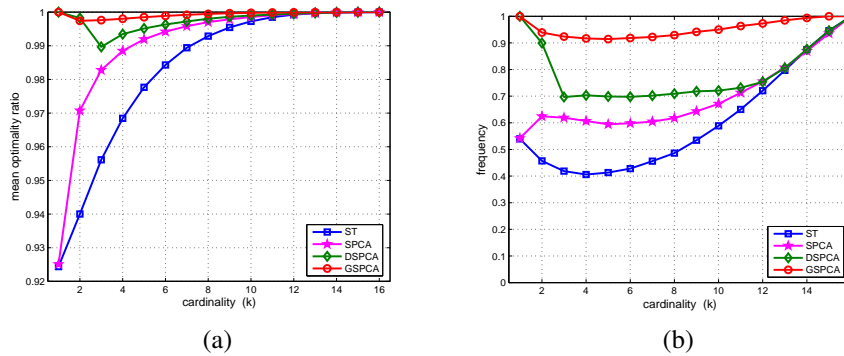

|     |     |
|:---:|:---:|
| (a) | (b) |

Figure 3: Monte Carlo summary statistics: **(a)** means of the distributions of optimality ratio (in Figure 2(b)) for all $k$ and **(b)** estimated probability of finding the *optimal* solution for each cardinality.

without the "fix" their variance curves are markedly diminished, as in Figure 2(a).

Figure 2(b) shows the histogram of the optimality ratio — *i.e.,* ratio of the captured to optimal variance — shown here at "half-sparsity" ($k = 8$, $n = 16$) from a typical MC run of 50,000 different covariances matrices. In order to view the (one-sided) tails of the distributions we have plotted the *log* of the histogram values. Figure 3(a) shows the corresponding mean values of the optimality ratio for all $k$. Among continuous algorithms, the SDP-based DSPCA was generally more effective (almost comparable to GSPCA). For the smaller matrices ($n < 10$), LARS-based SPCA matched DSPCA for all $k$. In terms of complexity and speed however, SPCA was about 40 times faster than DSPCA. But GSPCA was 30 times faster than SPCA. Finally, we note that even simple thresholding (ST), once enhanced with the variational renormalization, performs quite adequately despite its simplicity, as it captures at least 92% of the optimal variance, as seen in Figure 3(a).

Figure 3(b) shows an alternative but more revealing performance summary: the fraction of the (50,000) trials in which the optimal solution was actually found (essentially, the likelihood of "success"). This *all-or-nothing* performance measure elicits important differences between the algorithms. In practical terms, only GSPCA is capable of finding the optimal factor more than 90% of the time (*vs.* 70% for DSPCA). Naturally, *without* the variational "fix" (not shown) continuous algorithms rarely ever found the optimal solution.

## 4 Discussion

The contributions of this paper can be summarized as: (1) an *exact* variational formulation of sparse PCA, (2) requisite eigenvalue bounds, (3) a principled choice of $k$, (4) a simple renormalization "fix" for *any* continuous method, (5) fast and effective greedy search (GSPCA) and (6) a less efficient but *optimal* method (ESPCA). Surprisingly, simple thresholding of the principal eigenvector (ST) was shown to be rather effective, especially given the perceived "straw-man" it was considered to be. Naturally, its performance will vary with the effective rank (or "eigen-gap") of the covariance matrix. In fact, it is not hard to show that if $A$ is *exactly* rank-1, then ST is indeed an optimal strategy for all $k$. However, beyond such special cases, continuous methods can not ultimately be competitive with discrete algorithms without the variational renormalization "fix" in Section 2.2.

We should note that the somewhat remarkable effectiveness of GSPCA is not entirely unexpected and is supported by empirical observations in the combinatorial optimization literature: that greedy search with (sub)modular cost functions having the monotonicity property (*e.g.,* the variance curves $\tilde{v}(k)$) is known to produce good results [9]. In terms of quality of solutions, GSPCA consistently out-performed continuous algorithms, with run-times that were typically $O(10^2)$ faster than LARS-based SPCA and roughly $O(10^3)$ faster than SDP-based DSPCA (Matlab CPU times averaged over all $k$).

Nevertheless, we view discrete algorithms as complementary tools, especially since the leading continuous algorithms have distinct advantages. For example, with *very* high-dimensional datasets (*e.g.,* $n = 10,000$), Zou *et al.*'s LARS-based method is currently the only viable option, since it does not rely on computing or storing a *huge* covariance matrix. Although d'Aspremont *et al.* mention the possibility of solving "larger" systems much faster (using Nesterov's 1st-order method [3]), this would require a full matrix in memory (same as discrete algorithms). Still, their SDP formulation has an elegant *robustness* interpretation and can also be applied to *non-square* matrices (*i.e.,* for a sparse SVD).

### Acknowledgments

The authors would like to thank Karl Sjöstrand (DTU) for his customized code and helpful advice in using the LARS-SPCA toolbox [10] and Gert Lanckriet (Berkeley) for providing the Pit Props data.

## Footnotes

[1] We should note that the *multi-factor* version of Eq.(1) is *ill-posed* without additional constraints on basis orthogonality, cardinality, variable redundancy, ordinal rank and allocation of variance.

## References

[1] J. Cadima and I. Jolliffe. Loadings and correlations in the interpretation of principal components. *Applied Statistics*, 22:203–214, 1995.

[2] A. d'Aspremont. DSPCA Toolbox. http://www.princeton.edu/~aspremon/DSPCA.htm.

[3] A. d'Aspremont, L. El Ghaoui, M. I. Jordan, and G. R. G. Lanckriet. A Direct Formulation for Sparse PCA using Semidefinite Programming. In *Advances in Neural Information Processing Systems (NIPS)*. Vancouver, BC, December 2004.

[4] R. A. Horn and C. R. Johnson. *Matrix Analysis*. Cambridge Press, Cambridge, England, 1985.

[5] J. Jeffers. Two cases studies in the application of principal components. *Applied Statistics*, 16:225–236, 1967.

[6] I. T. Jolliffe and M. Uddin. A Modified Principal Component Technique based on the Lasso. *Journal of Computational and Graphical Statistics*, 12:531–547, 2003.

[7] C. Ko, J. Lee, and M. Queyranne. An Exact Algorithm for Maximum Entropy Sampling. *Operations Research*, 43(4):684–691, July-August 1995.

[8] G. McCabe. Principal variables. *Technometrics*, 26:137–144, 1984.

[9] G. L. Nemhauser and L. A. Wolsey. *Integer and Combinatorial Optimization*. John Wiley, New York, 1988.

[10] K. Sjöstrand. Matlab implementation of LASSO, LARS, the Elastic Net and SPCA. Informatics and Mathematical Modelling, Technical University of Denmark (DTU), 2005.

[11] J. F. Sturm. SeDuMi1.0x, a MATLAB Toolbox for Optimization over Symmetric Cones. *Optimization Methods and Software*, 11:625–653, 1999.

[12] R. Tibshirani. Regression shrinkage and selection via Lasso. *Journal of the Royal Statistical Society B*, 58:267–288, 1995.

[13] J. H. Wilkinson. *The Algebraic Eigenvalue Problem*. Clarendon Press, Oxford, England, 1965.

[14] H. Zou, T. Hastie, and R. Tibshirani. Sparse Principal Component Analysis. Technical Report, Statistics Department, Stanford University, 2004.
